# Improved Algorithms for Linear Stochastic Bandits

**Yasin Abbasi-Yadkori**
abbasiya@ualberta.ca
Dept. of Computing Science
University of Alberta

**Dávid Pál**
dpal@google.com
Dept. of Computing Science
University of Alberta

**Csaba Szepesvári**
szepesva@ualberta.ca
Dept. of Computing Science
University of Alberta

## Abstract

We improve the theoretical analysis and empirical performance of algorithms for the stochastic multi-armed bandit problem and the linear stochastic multi-armed bandit problem. In particular, we show that a simple modification of Auer's UCB algorithm (Auer, 2002) achieves with high probability constant regret. More importantly, we modify and, consequently, improve the analysis of the algorithm for the for linear stochastic bandit problem studied by Auer (2002), Dani et al. (2008), Rusmevichientong and Tsitsiklis (2010), Li et al. (2010). Our modification improves the regret bound by a logarithmic factor, though experiments show a vast improvement. In both cases, the improvement stems from the construction of smaller confidence sets. For their construction we use a novel tail inequality for vector-valued martingales.

## 1 Introduction

Linear stochastic bandit problem is a sequential decision-making problem where in each time step we have to choose an action, and as a response we receive a stochastic reward, expected value of which is an unknown linear function of the action. The goal is to collect as much reward as possible over the course of $n$ time steps. The precise model is described in Section 1.2.

Several variants and special cases of the problem exist differing on what the set of available actions is in each round. For example, the standard stochastic $d$-armed bandit problem, introduced by Robbins (1952) and then studied by Lai and Robbins (1985), is a special case of linear stochastic bandit problem where the set of available actions in each round is the standard orthonormal basis of $\mathbb{R}^d$. Another variant, studied by Auer (2002) under the name "linear reinforcement learning", and later in the context of web advertisement by Li et al. (2010), Chu et al. (2011), is a variant when the set of available actions changes from time step to time step, but has the same finite cardinality in each step. Another variant dubbed "sleeping bandits", studied by Kleinberg et al. (2008), is the case when the set of available actions changes from time step to time step, but it is always a subset of the standard orthonormal basis of $\mathbb{R}^d$. Another variant, studied by Dani et al. (2008), Abbasi-Yadkori et al. (2009), Rusmevichientong and Tsitsiklis (2010), is the case when the set of available actions does not change between time steps but the set can be an almost arbitrary, even infinite, bounded subset of a finite-dimensional vector space. Related problems were also studied by Abe et al. (2003), Walsh et al. (2009), Dekel et al. (2010).

In all these works, the algorithms are based on the same underlying idea—the *optimism-in-the-face-of-uncertainty* (OFU) principle. This is not surprising since they are solving almost the same problem. The OFU principle elegantly solves the exploration-exploitation dilemma inherent in the problem. The basic idea of the principle is to maintain a confidence set for the vector of coefficients of the linear function. In every round, the algorithm chooses an estimate from the confidence set and an action so that the predicted reward is maximized, i.e., estimate-action pair is chosen optimistically. We give details of the algorithm in Section 2.

Thus the problem reduces to the construction of confidence sets for the vector of coefficients of the linear function based on the action-reward pairs observed in the past time steps. This is not an easy problem, because the future actions are **not** independent of the actions taken in the past (since the algorithm's choices of future actions depend on the random confidence set constructed from past data). In fact, several authors (Auer, 2000, Li et al., 2010, Walsh et al., 2009) fell victim of making a mistake because they did not recognize this issue. Correct solutions require new martingale techniques which we provide here.

The smaller confidence sets one is able to construct, the better regret bounds one obtains for the resulting algorithm, and, more importantly, the better the algorithm performs empirically. With our new technique, we vastly reduce the size of the confidence sets of Dani et al. (2008) and Rusmevichientong and Tsitsiklis (2010). First, our confidence sets are valid uniformly over all time steps, which immediately saves $\log(n)$ factor by avoiding the otherwise needed union bound. Second, our confidence sets are "more empirical" in the sense that some worst-case quantities from the old bounds are replaced by empirical quantities that are always smaller, sometimes substantially. As a result, our experiments show an order-of-magnitude improvement over the CONFIDENCEBALL algorithm of Dani et al. (2008). To construct our confidence sets, we prove a new martingale tail inequality. The new inequality is derived using techniques from the theory of self-normalized processes (de la Peña et al., 2004, 2009).

Using our confidence sets, we modify the UCB algorithm (Auer, 2002) for the $d$-armed bandit problem and show that with probability $1 - \delta$, the regret of the modified algorithm is $O(d \log(1/\delta)/\Delta)$ where $\Delta$ is the difference between the expected rewards of the best and the second best action. In particular, note that the regret does not depend on $n$. This seemingly contradicts the result of Lai and Robbins (1985) who showed that the *expected* regret of any algorithm is at least $(\sum_{i \neq i_*} 1/D(p_j \mid p_{i_*}) - o(1)) \log n$ where $p_{i_*}$ and $p_i$ are the reward distributions of the optimal arm and arm $i$ respectively and $D$ is the Kullback-Leibler divergence. However, our algorithm receives $\delta$ as an input, and thus its expected regret depends on $\delta$. With $\delta = 1/n$ our algorithm has the same expected regret bound, $O((d \log n)/\Delta)$, as Auer (2002) has shown for UCB.

For the general linear stochastic bandit problem, we improve regret of the CONFIDENCEBALL algorithm of Dani et al. (2008). They showed that its regret is at most $O(d \log(n) \sqrt{n \log(n/\delta)})$ with probability at least $1 - \delta$. We modify their algorithm so that it uses our new confidence sets and we show that its regret is at most $O(d \log(n) \sqrt{n} + \sqrt{dn \log(n/\delta)})$ which is roughly improvement a multiplicative factor $\sqrt{\log(n)}$. Dani et al. (2008) prove also a problem dependent regret bound. Namely, they show that the regret of their algorithm is $O(\frac{d^2}{\Delta} \log(n/\delta) \log^2(n))$ where $\Delta$ is the "gap" as defined in (Dani et al., 2008). For our modified algorithm we prove an improved $O(\frac{\log(1/\delta)}{\Delta} (\log(n) + d \log \log n)^2)$ bound.

## 1.1 Notation

We use $\|x\|_p$ to denote the $p$-norm of a vector $x \in \mathbb{R}^d$. For a positive definite matrix $A \in \mathbb{R}^{d \times d}$, the weighted 2-norm of vector $x \in \mathbb{R}^d$ is defined by $\|x\|_A = \sqrt{x^\top A x}$. The inner product is denoted by $\langle \cdot, \cdot \rangle$ and the weighted inner-product $x^\top A y = \langle x, y \rangle_A$. We use $\lambda_{\min}(A)$ to denote the minimum eigenvalue of the positive definite matrix $A$. For any sequence $\{a_t\}_{t=0}^\infty$ we denote by $a_{i:j}$ the sub-sequence $a_i, a_{i+1}, \ldots, a_j$.

## 1.2 The Learning Model

In each round $t$, the learner is given a decision set $D_t \subseteq \mathbb{R}^d$ from which he has to choose an action $X_t$. Subsequently he observes reward $Y_t = \langle X_t, \theta_* \rangle + \eta_t$ where $\theta_* \in \mathbb{R}^d$ is an unknown parameter and $\eta_t$ is a random noise satisfying $\mathbf{E}[\eta_t \mid X_{1:t}, \eta_{1:t-1}] = 0$ and some tail-constraints, to be specified soon.

The goal of the learner is to maximize his total reward $\sum_{t=1}^n \langle X_t, \theta_* \rangle$ accumulated over the course of $n$ rounds. Clearly, with the knowledge of $\theta_*$, the optimal strategy is to choose in round $t$ the point $x_t^* = \operatorname{argmax}_{x \in D_t} \langle x, \theta_* \rangle$ that maximizes the reward. This strategy would accumulate total reward $\sum_{t=1}^n \langle x_t^*, \theta_* \rangle$. It is thus natural to evaluate the learner relative to this optimal strategy. The difference of the learner's total reward and the total reward of the optimal strategy is called the

```
for t := 1, 2, ... do
    (X_t, θ̃_t) = argmax_{(x,θ)∈D_t×C_{t-1}} ⟨x, θ⟩
    Play X_t and observe reward Y_t
    Update C_t
end for
```

Figure 1: OFUL ALGORITHM

*pseudo-regret* (Audibert et al., 2009) of the algorithm and it can be formally written as

$$R_n = \left( \sum_{t=1}^{n} \langle x_t^*, \theta_* \rangle \right) - \left( \sum_{t=1}^{n} \langle X_t, \theta_* \rangle \right) = \sum_{t=1}^{n} \langle x_t^* - X_t, \theta_* \rangle \ .$$

As compared to the regret, the pseudo-regret has the same expected value, but lower variance because the additive noise $\eta_t$ is removed. However, the omitted quantity is uncontrollable, hence we have no interest in including it in our results (the omitted quantity would also cancel, if $\eta_t$ was a sequence which is independently selected of $X_{1:t}$.) In what follows, for simplicity we use the word *regret* instead of the more precise pseudo-regret in connection to $R_n$.

The goal of the algorithm is to keep the regret $R_n$ as low as possible. As a bare minimum, we require that the algorithm is Hannan consistent, i.e., $R_n/n \to 0$ with probability one.

In order to obtain meaningful upper bounds on the regret, we will place assumptions on $\{D_t\}_{t=1}^{\infty}$, $\theta_*$ and the distribution of $\{\eta_t\}_{t=1}^{\infty}$. Roughly speaking, we will need to assume that $\{D_t\}_{t=1}^{\infty}$ lies in a bounded set. We elaborate on the details of the assumptions later in the paper.

However, we state the precise assumption on the noise sequence $\{\eta_t\}_{t=1}^{\infty}$ now. We will assume that $\eta_t$ is conditionally $R$-sub-Gaussian where $R \geq 0$ is a fixed constant. Formally, this means that

$$\forall \lambda \in \mathbb{R} \qquad \mathbf{E}\left[ e^{\lambda \eta_t} \mid X_{1:t}, \eta_{1:t-1} \right] \leq \exp\left( \frac{\lambda^2 R^2}{2} \right) \ .$$

The sub-Gaussian condition automatically implies that $\mathbf{E}[\eta_t \mid X_{1:t}, \eta_{1:t-1}] = 0$. Furthermore, it also implies that $\mathbf{Var}[\eta_t \mid F_t] \leq R^2$ and thus we can think of $R^2$ as the (conditional) variance of the noise. An example of $R$-sub-Gaussian $\eta_t$ is a zero-mean Gaussian noise with variance at most $R^2$, or a bounded zero-mean noise lying in an interval of length at most $2R$.

## 2 Optimism in the Face of Uncertainty

A natural and successful way to design an algorithm is the *optimism in the face of uncertainty principle* (OFU). The basic idea is that the algorithm maintains a confidence set $C_{t-1} \subseteq \mathbb{R}^d$ for the parameter $\theta_*$. It is required that $C_{t-1}$ can be calculated from $X_1, X_2, \ldots, X_{t-1}$ and $Y_1, Y_2, \ldots, Y_{t-1}$ and "with high probability" $\theta_*$ lies in $C_{t-1}$. The algorithm chooses an optimistic estimate $\widetilde{\theta}_t = \text{argmax}_{\theta \in C_{t-1}}(\max_{x \in D_t} \langle x, \theta \rangle)$ and then chooses action $X_t = \text{argmax}_{x \in D_t} \langle x, \widetilde{\theta}_t \rangle$ which maximizes the reward according to the estimate $\widetilde{\theta}_t$. Equivalently, and more compactly, the algorithm chooses the pair

$$(X_t, \widetilde{\theta}_t) = \underset{(x,\theta) \in D_t \times C_{t-1}}{\text{argmax}} \langle x, \theta \rangle \ ,$$

which *jointly* maximizes the reward. We call the resulting algorithm the OFUL ALGORITHM for "optimism in the face of uncertainty linear bandit algorithm". Pseudo-code of the algorithm is given in Figure 1.

The crux of the problem is the construction of the confidence sets $C_t$. This construction is the subject of the next section.

## 3 Self-Normalized Tail Inequality for Vector-Valued Martingales

Since the decision sets $\{D_t\}_{t=1}^{\infty}$ can be arbitrary, the sequence of actions $X_t \in D_t$ is arbitrary as well. Even if $\{D_t\}_{t=1}^{\infty}$ is "well-behaved", the selection rule that OFUL uses to choose $X_t \in D_t$

generates a sequence $\{X_t\}_{t=1}^{\infty}$ with complicated stochastic dependencies that are hard to handle. Therefore, for the purpose of deriving confidence sets it is easier to drop any assumptions on $\{X_t\}_{t=1}^{\infty}$ and pursue a more general result.

If we consider the $\sigma$-algebra $F_t = \sigma(X_1, X_2, \ldots, X_{t+1}, \eta_1, \eta_2, \ldots, \eta_t)$ then $X_t$ becomes $F_{t-1}$-measurable and $\eta_t$ becomes $F_t$-measurable. Relaxing this a little bit, we can assume that $\{F_t\}_{t=0}^{\infty}$ is any filtration of $\sigma$-algebras such that for any $t \geq 1$, $X_t$ is $F_{t-1}$-measurable and $\eta_t$ is $F_t$-measurable and therefore $Y_t = \langle X_t, \theta_* \rangle + \eta_t$ is $F_t$-measurable. This is the setup we consider for derivation of the confidence sets.

The sequence $\{S_t\}_{t=0}^{\infty}$, $S_t = \sum_{s=1}^{t} \eta_t X_t$, is a martingale with respect $\{F_t\}_{t=0}^{\infty}$ which happens to be crucial for the construction of the confidence sets for $\theta_*$. The following theorem shows that with high probability the martingale stays close to zero. Its proof is given in Appendix A

**Theorem 1** (Self-Normalized Bound for Vector-Valued Martingales). *Let $\{F_t\}_{t=0}^{\infty}$ be a filtration. Let $\{\eta_t\}_{t=1}^{\infty}$ be a real-valued stochastic process such that $\eta_t$ is $F_t$-measurable and $\eta_t$ is conditionally $R$-sub-Gaussian for some $R \geq 0$ i.e.*

$$\forall \lambda \in \mathbb{R} \qquad \mathbf{E}\left[e^{\lambda \eta_t} \mid F_{t-1}\right] \leq \exp\left(\frac{\lambda^2 R^2}{2}\right) .$$

*Let $\{X_t\}_{t=1}^{\infty}$ be an $\mathbb{R}^d$-valued stochastic process such that $X_t$ is $F_{t-1}$-measurable. Assume that $V$ is a $d \times d$ positive definite matrix. For any $t \geq 0$, define*

$$\overline{V}_t = V + \sum_{s=1}^{t} X_s X_s^{\top} \qquad\qquad S_t = \sum_{s=1}^{t} \eta_s X_s .$$

*Then, for any $\delta > 0$, with probability at least $1 - \delta$, for all $t \geq 0$,*

$$\|S_t\|_{\overline{V}_t^{-1}}^2 \leq 2R^2 \log\left(\frac{\det(\overline{V}_t)^{1/2} \det(V)^{-1/2}}{\delta}\right) .$$

Note that the deviation of the martingale $\|S_t\|_{\overline{V}_t^{-1}}^2$ is measured by the norm weighted by the matrix $\overline{V}_t^{-1}$ which is itself derived from the martingale, hence the name "self-normalized bound".

## 4 Construction of Confidence Sets

Let $\widehat{\theta}_t$ be the $\ell^2$-regularized least-squares estimate of $\theta_*$ with regularization parameter $\lambda > 0$:

$$\widehat{\theta}_t = (\mathbf{X}_{1:t}^{\top} \mathbf{X}_{1:t} + \lambda I)^{-1} \mathbf{X}_{1:t}^{\top} \mathbf{Y}_{1:t} \tag{1}$$

where $\mathbf{X}_{1:t}$ is the matrix whose rows are $X_1^{\top}, X_2^{\top}, \ldots, X_t^{\top}$ and $\mathbf{Y}_{1:t} = (Y_1, \ldots, Y_t)^{\top}$. The following theorem shows that $\theta_*$ lies with high probability in an ellipsoid with center at $\widehat{\theta}_t$. Its proof can be found in Appendix B.

**Theorem 2** (Confidence Ellipsoid). *Assume the same as in Theorem 1, let $V = I\lambda$, $\lambda > 0$, define $Y_t = \langle X_t, \theta_* \rangle + \eta_t$ and assume that $\|\theta_*\|_2 \leq S$. Then, for any $\delta > 0$, with probability at least $1 - \delta$, for all $t \geq 0$, $\theta_*$ lies in the set*

$$C_t = \left\{ \theta \in \mathbb{R}^d \ : \ \left\|\widehat{\theta}_t - \theta\right\|_{\overline{V}_t} \leq R\sqrt{2\log\left(\frac{\det(\overline{V}_t)^{1/2} \det(\lambda I)^{-1/2}}{\delta}\right)} + \lambda^{1/2} S \right\} .$$

*Furthermore, if for all $t \geq 1$, $\|X_t\|_2 \leq L$ then with probability at least $1 - \delta$, for all $t \geq 0$, $\theta_*$ lies in the set*

$$C_t' = \left\{ \theta \in \mathbb{R}^d \ : \ \left\|\widehat{\theta}_t - \theta\right\|_{\overline{V}_t} \leq R\sqrt{d\log\left(\frac{1 + tL^2/\lambda}{\delta}\right)} + \lambda^{1/2} S \right\} .$$

The above bound could be compared with a similar bound of Dani et al. (2008) whose bound, under identical conditions, states that (with appropriate initialization) with probability $1 - \delta$,

$$\text{for all } t \text{ large enough} \quad \left\| \widehat{\theta}_t - \theta_* \right\|_{\overline{V}_t} \leq R \max \left\{ \sqrt{128\, d \log(t) \, \log\left(\frac{t^2}{\delta}\right)}, \frac{8}{3} \log\left(\frac{t^2}{\delta}\right) \right\} , \quad (2)$$

where large enough means that $t$ satisfies $0 < \delta < t^2 e^{-1/16}$. Denote by $\sqrt{\beta_t(\delta)}$ the right-hand side in the last bound. The restriction on $t$ comes from the fact that $\beta_t(\delta) \geq 2d(1 + 2\log(t))$ is needed in the proof of the last inequality of their Theorem 5.

On the other hand, Rusmevichientong and Tsitsiklis (2010) proved that for any *fixed* $t \geq 2$, for any $0 < \delta < 1$, with probability at least $1 - \delta$,

$$\left\| \widehat{\theta}_t - \theta_* \right\|_{\overline{V}_t} \leq 2\,\kappa^2 R \sqrt{\log t} \, \sqrt{d \, \log(t) + \log(1/\delta)} + \lambda^{1/2} S \,,$$

where $\kappa = \sqrt{3 + 2\log((L^2 + \text{trace}(V))/\lambda)}$. To get a uniform bound one can use a union bound with $\delta_t = \delta/t^2$. Then $\sum_{t=2}^{\infty} \delta_t = \delta(\frac{\pi^2}{6} - 1) \leq \delta$. This thus gives that for any $0 < \delta < 1$, with probability at least $1 - \delta$,

$$\forall t \geq 2, \quad \left\| \widehat{\theta}_t - \theta_* \right\|_{\overline{V}_t} \leq 2\kappa^2 R \sqrt{\log t} \, \sqrt{d \, \log(t) + \log(t^2/\delta)} + \lambda^{1/2} S \,,$$

This is tighter than (2), but is still lagging behind the result of Theorem 2. Note that the new confidence set seems to require the computation of a determinant of a matrix, a potentially expensive step. However, one can speed up the computation by using the matrix determinant lemma, exploiting that the matrix whose determinant is needed is obtained via a rank-one update (cf. the proof of Lemma 11 in the Appendix). This way, the determinant can be kept up-to-date with linear time computation.

## 5 Regret Analysis of the OFUL ALGORITHM

We now give a bound on the regret of the OFUL algorithm when run with confidence sets $C_n$ constructed in Theorem 2 in the previous section. We will need to assume that expected rewards are bounded. We can view this as a bound on $\theta_*$ and the bound on the decision sets $D_t$. The next theorem states a bound on the regret of the algorithm. Its proof can be found in Appendix C.

**Theorem 3** (The regret of the OFUL algorithm). *Assume that for all $t$ and all $x \in D_t$, $\langle x, \theta_* \rangle \in [-1, 1]$. Then, with probability at least $1 - \delta$, the regret of the* OFUL *algorithm satisfies*

$$\forall n \geq 0, \quad R_n \leq 4\sqrt{nd \log(\lambda + nL/d)} \left( \lambda^{1/2} S + R\sqrt{2\log(1/\delta) + d\log(1 + nL/(\lambda d))} \right) \,.$$

Figure 2 shows the experiments with the new confidence set. The regret of OFUL is significantly better compared to the regret of CONFIDENCEBALL of Dani et al. (2008). The figure also shows a version of the algorithm that has a similar regret to the algorithm with the new bound, but spends about 350 times less computation in this experiment. Next, we explain how we can achieve this computation saving.

### 5.1 Saving Computation

In this section, we show that we essentially need to recompute $\widetilde{\theta}_t$ only $O(\log n)$ times up to time $n$ and hence saving computations.[1] The idea is to recompute $\widetilde{\theta}_t$ whenever $\det(V_t)$ increases by a constant factor $(1 + C)$. We call the resulting algorithm the RARELY SWITCHING OFUL algorithm and its pseudo-code is given in Figure 3. As the next theorem shows its regret bound is essentially the same as the regret for OFUL.

**Theorem 4.** *Under the same assumptions as in Theorem 3, with probability at least $1 - \delta$, for all $n \geq 0$, the regret of the* RARELY SWITCHING OFUL ALGORITHM *satisfies*

$$R_n \leq 4\sqrt{(1 + C)nd \log\left(\lambda + \frac{nL}{d}\right)} \left\{ \sqrt{\lambda} S + R\sqrt{d\log\left(1 + \frac{nL}{\lambda d}\right) + 2\log \frac{1}{\delta}} \right\} + 4\sqrt{d\log \frac{n}{d}} \,.$$

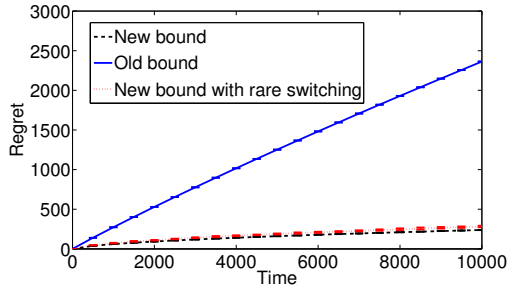

Figure 2: The application of the new bound to a linear bandit problem. A 2-dimensional linear bandit, where the parameters vector and the actions are from the unit ball. The regret of OFUL is significantly better compared to the regret of CONFIDENCEBALL of Dani et al. (2008). The noise is a zero mean Gaussian with standard deviation $\sigma = 0.1$. The probability that confidence sets fail is $\delta = 0.0001$. The experiments are repeated 10 times.

---

**Input:** Constant $C > 0$
$\tau = 1$ {This is the last time step that we changed $\widetilde{\theta}_t$}
**for** $t := 1, 2, \ldots$ **do**
    **if** $\det(V_t) > (1 + C)\det(V_\tau)$ **then**
        $(X_t, \widetilde{\theta}_t) = \operatorname{argmax}_{(x,\theta) \in D_t \times C_{t-1}} \langle \theta, x \rangle.$
        $\tau = t.$
    **end if**
    $X_t = \operatorname{argmax}_{x \in D_t} \left\langle \widetilde{\theta}_\tau, x \right\rangle.$
    Play $X_t$ and observe reward $Y_t$.
**end for**

---

Figure 3: The RARELY SWITCHING OFUL ALGORITHM

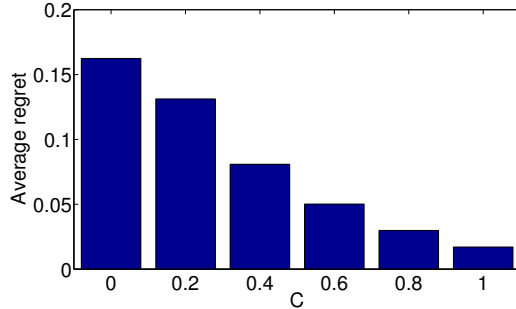

Figure 4: Regret against computation. We fixed the number of times the algorithm is allowed to update its action in OFUL. For larger values of $C$, the algorithm changes action less frequently, hence, will play for a longer time period. The figure shows the average regret obtained during the given time periods for the different values of $C$. Thus, we see that by increasing $C$, one can actually lower the average regret per time step for a given fixed computation budget.

The proof of the theorem is given in Appendix D. Figure 4 shows a simple experiment with the RARELY SWITCHING OFUL ALGORITHM.

## 5.2 Problem Dependent Bound

Let $\Delta_t$ be the "gap" at time step $t$ as defined in (Dani et al., 2008). (Intuitively, $\Delta_t$ is the difference between the rewards of the best and the "second best" action in the decision set $D_t$.) We consider

the smallest gap $\bar{\Delta}_n = \min_{1 \le t \le n} \Delta_t$. This includes the case when the set $D_t$ is the same polytope in every round or the case when $D_t$ is finite.

The regret of OFUL can be upper bounded in terms of $(\bar{\Delta}_n)_n$ as follows.

**Theorem 5.** *Assume that $\lambda \ge 1$ and $\|\theta_*\|_2 \le S$ where $S \ge 1$. With probability at least $1 - \delta$, for all $n \ge 1$, the regret of the* OFUL *satisfies*

$$R_n \le \frac{16R^2\lambda S^2}{\bar{\Delta}_n}\Big(\log(Ln) + (d-1)\log\frac{64R^2\lambda S^2 L}{\bar{\Delta}_n^2}$$
$$+ 2(d-1)\log\left(d\log\frac{d\lambda + nL^2}{d} + 2\log(1/\delta)\right) + 2\log(1/\delta)\Big)^2 .$$

The proof of the theorem can be found in the Appendix E.

The problem dependent regret of (Dani et al., 2008) scales like $O(\frac{d^2}{\Delta}\log^3 n)$, while our bound scales like $O(\frac{1}{\Delta}(\log^2 n + d\log n + d^2\log\log n))$, where $\Delta = \inf_n \bar{\Delta}_n$.

# 6 Multi-Armed Bandit Problem

In this section we show that a modified version of UCB has with high probability constant regret.

Let $\mu_i$ be the expected reward of action $i = 1, 2, \ldots, d$. Let $\mu_* = \max_{1 \le i \le d} \mu_i$ be the expected reward of the best arm, and let $\Delta_i = \mu_* - \mu_i$, $i = 1, 2, \ldots, d$, be the "gaps" with respect to the best arm. We assume that if we choose action $I_t$ in round $t$ we obtain reward $\mu_{I_t} + \eta_t$. Let $N_{i,t}$ denote the number of times that we have played action $i$ up to time $t$, and $\overline{X}_{i,t}$ denote the average of the rewards received by action $i$ up to time $t$. We construct confidence intervals for the expected rewards $\mu_i$ based on $\overline{X}_{i,t}$ in the following lemma. (The proof can be found in the Appendix F.)

**Lemma 6** (Confidence Intervals)**.** *Assuming that the noise $\eta_t$ is conditionally 1-sub-Gaussian. With probability at least $1 - \delta$,*

$$\forall i \in \{1, 2, \ldots, d\}, \; \forall t \ge 0 \qquad |\overline{X}_{i,t} - \mu_i| \le c_{i,t} ,$$

*where*

$$c_{i,t} = \sqrt{\frac{(1 + N_{i,t})}{N_{i,t}^2}\left(1 + 2\log\left(\frac{d(1 + N_{i,t})^{1/2}}{\delta}\right)\right)} . \tag{3}$$

Using these confidence intervals, we modify the UCB algorithm of Auer et al. (2002) and change the action selection rule accordingly. Hence, at time $t$, we choose the action

$$I_t = \operatorname*{argmax}_i \overline{X}_{i,t} + c_{i,t}. \tag{4}$$

We call this algorithm UCB($\delta$).

The main difference between UCB($\delta$) and UCB is that the length of confidence interval $c_{i,t}$ depends neither on $n$, nor on $t$. This allows us to prove the following result that the regret of UCB($\delta$) is constant. (The proof can be found in the Appendix G.)

**Theorem 7** (Regret of UCB($\delta$))**.** *Assume that the noise $\eta_t$ is conditionally 1-sub-Gaussian, with probability at least $1 - \delta$, the total regret of the UCB($\delta$) is bounded as*

$$R_n \le \sum_{i:\Delta_i > 0}\left(3\Delta_i + \frac{16}{\Delta_i}\log\frac{2d}{\Delta_i\delta}\right) .$$

Lai and Robbins (1985) prove that for any suboptimal arm $j$,

$$\mathbf{E}\,N_{i,t} \ge \frac{\log t}{D(p_j, p_*)},$$

where, $p_*$ and $p_j$ are the reward density of the optimal arm and arm $j$ respectively, and $D$ is the KL-divergence. This lower bound does not contradict Theorem 7, as Theorem 7 only states a high

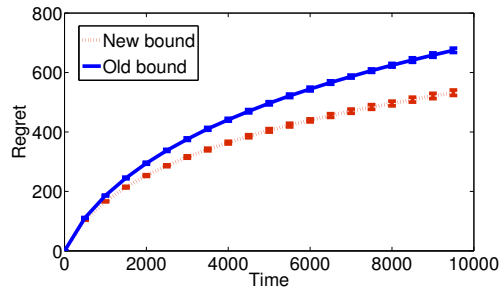

Figure 5: The regret of UCB($\delta$) against-time when it uses either the confidence bound based on Hoeffding's inequality, or the bound in (3). The results are shown for a 10-armed bandit problem, where the mean value of each arm is fixed to some values in $[0, 1]$. The regret of UCB($\delta$) is improved with the new bound. The noise is a zero-mean Gaussian with standard deviation $\sigma = 0.1$. The value of $\delta$ is set to $0.0001$. The experiments are repeated 10 times and the average is shown, together with the error bars.

probability upper bound for the regret. Note that UCB($\delta$) takes delta as its input. Because with probability $\delta$, the regret in time $t$ can be $t$, on expectation, the algorithm might have a regret of $t\delta$. Now if we select $\delta = 1/t$, then we get $O(\log t)$ upper bound on the expected regret.

If one is interested in an average regret result, then, with slight modification of the proof technique one can obtain an identical result to what Auer et al. (2002) proves.

Figure 5 shows the regret of UCB($\delta$) when it uses either the confidence bound based on Hoeffding's inequality, or the bound in (3). As can be seen, the regret of UCB($\delta$) is improved with the new bound.

Coquelin and Munos (2007), Audibert et al. (2009) prove similar high-probability constant regret bounds for variations of the UCB algorithm. Compared to their bounds, our bound is tighter thanks to that with the new self-normalized tail inequality we can avoid one union bound. The improvement can also be seen in experiment as the curve that we get for the performance of the algorithm of Coquelin and Munos (2007) is almost exactly the same as the curve that is labeled "Old Bound" in Figure 5.

## 7   Conclusions

In this paper, we showed how a novel tail inequality for vector-valued martingales allows one to improve both the theoretical analysis and empirical performance of algorithms for various stochastic bandit problems. In particular, we show that a simple modification of Auer's UCB algorithm (Auer, 2002) achieves with high probability constant regret. Further, we modify and improve the analysis of the algorithm for the for linear stochastic bandit problem studied by Auer (2002), Dani et al. (2008), Rusmevichientong and Tsitsiklis (2010), Li et al. (2010). Our modification improves the regret bound by a logarithmic factor, though experiments show a vast improvement, stemming from the construction of smaller confidence sets. To our knowledge, ours is the first, theoretically well-founded algorithm, whose performance is *practical* for this latter problem. We also proposed a novel variant of the algorithm with which we can save a large amount of computation without sacrificing performance.

We expect that the novel tail inequality will also be useful in a number of other situations thanks to its self-normalized form and that it holds for stopped martingales and thus can be used to derive bounds that hold uniformly in time. In general, the new inequality can be used to improve deviation bounds which use a union bound (over time). Since many modern machine learning techniques rely on having tight high-probability bounds, we expect that the new inequality will find many applications. Just to mention a few examples, the new inequality could be used to improve the computational complexity of the HOO algorithm Bubeck et al. (2008) (when it is used with a fixed $\delta$, by avoiding union bounds, or the need to know the horizon, or the doubling trick) or to improve the bounds derived by Garivier and Moulines (2008) for UCB for changing environments, or the stopping rules and racing algorithms of Mnih et al. (2008).

## Footnotes

[1]Note this is very different than the common "doubling trick" in online learning literature. The doubling is used to cope with a different problem. Namely, the problem when the time horizon $n$ is unknown ahead of time.

# References

Y. Abbasi-Yadkori, A. Antos, and Cs. Szepesvári. Forced-exploration based algorithms for playing in stochastic linear bandits. In *COLT Workshop on On-line Learning with Limited Feedback*, 2009.

N. Abe, A. W. Biermann, and P. M. Long. Reinforcement learning with immediate rewards and linear hypotheses. *Algorithmica*, 37:263293, 2003.

A. Antos, V. Grover, and Cs. Szepesvári. Active learning in heteroscedastic noise. *Theoretical Computer Science*, 411(29-30):2712–2728, 2010.

J.-Y. Audibert, R. Munos, and Csaba Szepesvári. Exploration-exploitation tradeoff using variance estimates in multi-armed bandits. *Theoretical Computer Science*, 410(19):1876–1902, 2009.

P. Auer. Using upper confidence bounds for online learning. In *FOCS*, pages 270–279, 2000.

P. Auer. Using confidence bounds for exploitation-exploration trade-offs. *JMLR*, 2002.

P. Auer, N. Cesa-Bianchi, and P. Fischer. Finite time analysis of the multiarmed bandit problem. *Machine Learning*, 47(2-3):235–256, 2002.

S. Bubeck, R. Munos, G. Stoltz, and Cs. Szepesvári. Online optimization in X-armed bandits. In *NIPS*, pages 201–208, 2008.

N. Cesa-Bianchi and G. Lugosi. *Prediction, Learning, and Games*. 2006.

W. Chu, L. Li, L. Reyzin, and R. E. Schapire. Contextual bandits with linear payoff functions. In *AISTATS*, 2011.

P.-A. Coquelin and R. Munos. Bandit algorithms for tree search. In *UAI*, 2007.

V. Dani, T. P. Hayes, and S. M. Kakade. Stochastic linear optimization under bandit feedback. In Rocco Servedio and Tong Zhang, editors, *COLT*, pages 355–366, 2008.

V. H. de la Peña, M. J. Klass, and T. L. Lai. Self-normalized processes: exponential inequalities, moment bounds and iterated logarithm laws. *Annals of Probability*, 32(3):1902–1933, 2004.

V. H. de la Peña, T. L. Lai, and Q.-M. Shao. *Self-normalized processes: Limit theory and Statistical Applications*. Springer, 2009.

O. Dekel, C. Gentile, and K. Sridharan. Robust selective sampling from single and multiple teachers. In *COLT*, 2010.

D. A. Freedman. On tail probabilities for martingales. *The Annals of Probability*, 3(1):100–118, 1975.

A. Garivier and E. Moulines. On upper-confidence bound policies for non-stationary bandit problems. Technical report, LTCI, 2008.

R. Kleinberg, A. Niculescu-Mizil, and Y. Sharma. Regret bounds for sleeping experts and bandits. *Machine learning*, pages 1–28, 2008.

T. L. Lai and H. Robbins. Asymptotically efficient adaptive allocation rules. *Advances in Applied Mathematics*, 6:4–22, 1985.

T. L. Lai and C. Z. Wei. Least squares estimates in stochastic regression models with applications to identification and control of dynamic systems. *The Annals of Statistics*, 10(1):154–166, 1982.

T. L. Lai, H. Robbins, and C. Z. Wei. Strong consistency of least squares estimates in multiple regression. *Proceedings of the National Academy of Sciences*, 75(7):3034–3036, 1979.

L. Li, W. Chu, J. Langford, and R. E. Schapire. A contextual-bandit approach to personalized news article recommendation. In *Proceedings of the 19th International Conference on World Wide Web (WWW 2010)*, pages 661–670. ACM, 2010.

V. Mnih, Cs. Szepesvári, and J.-Y. Audibert. Empirical Bernstein stopping. pages 672–679, 2008.

H. Robbins. Some aspects of the sequential design of experiments. *Bulletin of the American Mathematical Society*, 58:527–535, 1952.

P. Rusmevichientong and J. N. Tsitsiklis. Linearly parameterized bandits. *Mathematics of Operations Research*, 35(2):395–411, 2010.

G. W. Stewart and J.-G. Sun. *Matrix Perturbation Theory*. Academic Press, 1990.

T. J. Walsh, I. Szita, C. Diuk, and M. L. Littman. Exploring compact reinforcement-learning representations with linear regression. In *UAI*, pages 591–598. AUAI Press, 2009.

